# Learning Many Related Tasks at the Same Time With Backpropagation

**Rich Caruana**
School of Computer Science
Carnegie Mellon University
Pittsburgh, PA 15213
caruana@cs.cmu.edu

## Abstract

Hinton [6] proposed that generalization in artificial neural nets should improve if nets learn to represent the domain's underlying regularities. Abu-Mustafa's *hints* work [1] shows that the *outputs* of a backprop net can be used as *inputs* through which domain-specific information can be given to the net. We extend these ideas by showing that a backprop net learning many related tasks at the same time can use these tasks as inductive bias for each other and thus learn better. We identify five mechanisms by which multitask backprop improves generalization and give empirical evidence that multitask backprop generalizes better in real domains.

## 1  INTRODUCTION

You and I rarely learn things one at a time, yet we often ask our programs to—it *must* be easier to learn things one at a time than to learn many things at once. Maybe not. The things you and I learn are related in many ways. They are processed by the same sensory apparatus, controlled by the same physical laws, derived from the same culture, ... Perhaps it is the *similarity* between the things we learn that helps us learn so well. What happens when a net learns many related functions at the same time? Will the extra information in the teaching signal of the related tasks help it learn better?

Section 2 describes five mechanisms that improve generalization in backprop nets trained simultaneously on related tasks. Section 3 presents empirical results from a road-following domain and an object-recognition domain where backprop with multiple tasks improves generalization 10–40%. Section 4 briefly discusses when and how to use multitask backprop. Section 5 cites related work and Section 6 outlines directions for future work.

## 2   MECHANISMS OF MULTITASK BACKPROP

We identified five mechanisms that improve generalization in backprop nets trained simultaneously on multiple related tasks. The mechanisms all derive from the summing of error gradient terms at the hidden layer from the different tasks. Each exploits a different relationship between the tasks.

### 2.1   Data Amplification

Data amplification is an *effective* increase in sample size due to extra information in the training signal of related tasks. There are two types of data amplification.

#### 2.1.1   Statistical Data Amplification

Statistical amplification, occurs when there is noise in the training signals. Consider two tasks, $T$ and $T'$, with independent noise added to their training signals, that both benefit from computing a feature $F$ of the inputs. A net learning both $T$ and $T'$ can, if it recognizes that the two tasks share $F$, use the two training signals to learn $F$ better by averaging $F$ through the noise. The simplest case is when $T = T'$, i.e., when the two outputs are independently corrupted versions of the same signal.

#### 2.1.2   Blocking Data Amplification

The 2nd form of data amplification occurs even if there is no noise. Consider two tasks, $T$ and $T'$, that use a common feature $F$ computable from the inputs, but each uses $F$ for different training patterns. A simple example is $T = A\,OR\,F$ and $T' = NOT(A)\,OR\,F$. $T$ uses $F$ when $A = 0$ and provides no information about $F$ when $A = 1$. Conversely, $T'$ provides information about $F$ only when $A = 1$. A net learning just $T$ gets information about $F$ only on training patterns for which $A = 0$, but is *blocked* when $A = 1$. But a net learning both $T$ and $T'$ at the same time gets information about $F$ on every training pattern; it is never blocked. *It does not see more training patterns, it gets more information for each pattern.* If the net learning both tasks recognizes the tasks share $F$, it will see a larger sample of $F$. Experiments with blocked functions like $T$ and $T'$ (where $F$ is a hard but learnable function of the inputs such as parity) indicate backprop does learn common subfeatures better due to the larger effective sample size.

### 2.2   Attribute Selection

Consider two tasks, $T$ and $T'$, that use a common subfeature $F$. Suppose there are many inputs to the net, but $F$ is a function of only a few of the inputs. A net learning $T$ will, if there is limited training data and/or significant noise, have difficulty distinguishing inputs relevant to $F$ from those irrelevant to it. A net learning both $T$ and $T'$, however, will better select the attributes relevant to $F$ because data amplification provides better training signals for $F$ and that allows it to better determine which inputs to use to compute $F$. (Note: data amplification occurs even when there is no attribute selection problem. Attribute selection is a consequence of data amplification that makes data amplification work better when a selection problem exists.) We detect attribute selection by looking for connections to relevant inputs that grow stronger compared to connections for irrelevant inputs when multiple tasks are trained on the net.

## 2.3 Eavesdropping

Consider a feature $F$, useful to tasks, $T$ and $T'$, that is easy to learn when learning $T$, but difficult to learn when learning $T'$ because $T'$ uses $F$ in a more complex way. A net learning $T$ will learn $F$, but a net learning just $T'$ may not. If the net learning $T'$ also learns $T$, $T'$ can *eavesdrop* on the hidden layer learned for $T$ (e.g., $F$) and thus learn better. Moreover, once the connection is made between $T'$ and the evolving representation for $F$, the extra information from $T'$ about $F$ will help the net learn $F$ better via the other mechanisms. The simplest case of eavesdropping is when $T = F$. Abu-Mostafa calls these *catalytic hints*[1]. In this case the net is being told explicitly to learn a feature $F$ that is useful to the main task. Eavesdropping sometimes causes non-monotonic generalization curves for the tasks that eavesdrop on other tasks. This happens when the eavesdropper begins to overtrain, but then finds something useful learned by another task, and begins to perform better as it starts using this new information.

## 2.4 Representation Bias

Because nets are initialized with random weights, backprop is a stochastic search procedure; multiple runs rarely yield identical nets. Consider the set of all nets (for fixed architecture) learnable by backprop for task $T$. Some of these generalize better than others because they better "represent" the domain's regularities. Consider one such regularity, $F$, learned differently by the different nets. Now consider the set of all nets learnable by backprop for another task $T'$ that also learns regularity $F$. If $T$ and $T'$ are both trained on one net and the net recognizes the tasks share $F$, search will be biased towards representations of $F$ near the intersection of what would be learned for $T$ or $T'$ alone. We conjecture that representations of $F$ near this intersection often better capture the true regularity of $F$ because they satisfy more than one task from the domain.

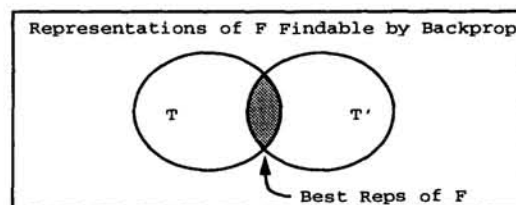

A form of representation bias that is easier to experiment with occurs when the representations for $F$ sampled by the two tasks are different minima. Suppose there are two minima, $A$ and $B$, a net can find for task $T$. Suppose a net learning task $T'$ also has two minima, $A$ and $C$. Both share the minima at $A$ (i.e., both would perform well if the net entered that region of weight space), but do not overlap at $B$ and $C$. We ran two experiments. In the first, we selected the minima so that nets trained on $T$ alone are equally likely to find $A$ or $B$, and nets trained on $T'$ alone are equally likely to find $A$ or $C$. Nets trained on both $T$ and $T'$ usually fall into $A$ for both tasks.[1] *Tasks prefer hidden layer representations that other tasks prefer.*

In the second experiment we selected the minima so that $T$ has a strong preference

for $B$ over $A$: a net trained on $T$ always falls into $B$. $T'$, however, still has no preference between $A$ or $C$. When both $T$ and $T'$ are trained on one net, $T$ falls into $B$ as expected: the bias from $T'$ is unable to pull it to $A$. Surprisingly, $T'$ usually falls into $C$, the minima it does not share with $T$! $T$ creates a "tide" in the hidden layer representation towards $B$ that flows away from $A$. $T'$ has no preference for $A$ or $C$, but is subject to the tide created by $T$. Thus $T'$ usually falls into $C$; it would have to fight the tide from $T$ to fall into $A$. *Tasks prefer NOT to use hidden layer representations that other tasks prefer NOT to use.*

### 2.5  How the Mechanisms are Related

The "tide" mentioned while discussing representation bias results from the aggregation of error gradients from multiple tasks at the hidden layer. It is what makes the five mechanisms tick. It biases the search trajectory towards better performing regions of weight space. Because the mechanisms arise from the same underlying cause, it easy for them to act in concert. Their combined effect can be substantial.

Although the mechanisms all derive from gradient summing, they are not the same. Each emphasizes a different relationship between tasks and has different effects on what is learned. Changes in architecture, representation, and the learning procedure affect the mechanisms in different ways. One particularly noteworthy difference between the mechanisms is that if there are minima, representation bias affects learning even with infinite sample size. The other mechanisms work only with finite sample size: data amplification (and thus attribute selection) and eavesdropping are beneficial only when the sample size is too small for the training signal for one task to provide enough information to the net for it to learn good models.

## 3  EMPIRICAL RESULTS

Experiments on carefully crafted test problems verify that each of the mechanisms can work.[2] These experiments, however, do not indicate how effective multitask backprop is on real problems: tweaking the test problems alters the size of the effects. Rather than present results for contrived problems, we present a more convincing demonstration of multitask backprop by testing it on two realistic domains.

### 3.1  1D-ALVINN

1D-ALVINN uses a road image simulator developed by Pomerleau. It was modified to generate 1-D road images comprised of a single 32-pixel horizontal scan line instead of the original 2-D 30x32-pixel image. This was done to speed learning to allow thorough experimentation. 1D-ALVINN retains much of the complexity of the original 2-D domain—the complexity lost is road curvature and that due to the smaller input (960 pixels vs. 32 pixels). The principle task in 1D-ALVINN is to predict steering direction. Eight additional tasks were used for multitask backprop:

- whether the road is one or two lanes
- location of left edge of road
- location of road center
- intensity of region bordering road

- location of centerline (2-lane roads only)
- location of right edge of road
- intensity of road surface
- intensity of centerline (2-lane roads only)

Table 1 shows the performance of single and multitask backprop (STB and MTB, respectively) on 1D-ALVINN using nets with one hidden layer. The MTB net has 32 inputs, 16 hidden units, and 9 outputs. The 36 STB nets have 32 inputs, 2, 4, 8 or 16 hidden units, and 1 output. A similar experiment using nets with 2 hidden layers containing 2, 4, 8, 16, or 32 hidden units per layer for STB and 32 hidden units per layer for MTB yielded comparable results. The size of the MTB nets is not optimized in either experiment.

Table 1: Performance of STB and MTB with One Hidden Layer on 1D-ALVINN

| TASK | ROOT-MEAN SQUARED ERROR ON TEST SET | | | | | | |
| | Single Task Backprop | | | | MTB | % Change | % Change |
| | 2HU | 4HU | 8HU | 16HU | 16HU | Best STB | Mean STB |
|---|---|---|---|---|---|---|---|
| 1 or 2 Lanes | .201 | .209 | .207 | *.178* | .156 | 14.1 | 27.4 |
| Left Edge | *.069* | .071 | .073 | .073 | .062 | 11.3 | 15.3 |
| Right Edge | .076 | .062 | .058 | *.056* | .051 | 9.8 | 23.5 |
| Line Center | .153 | *.152* | .152 | .152 | .151 | 0.7 | 0.8 |
| Road Center | .038 | *.037* | .039 | .042 | .034 | 8.8 | 14.7 |
| Road Greylevel | *.054* | .055 | .055 | .054 | .038 | 42.1 | 43.4 |
| Edge Greylevel | *.037* | .038 | .039 | .038 | .038 | -2.6 | 0.0 |
| Line Greylevel | .054 | .054 | *.054* | .054 | .054 | 0.0 | 0.0 |
| Steering | .093 | *.069* | .087 | .072 | .058 | 19.0 | 38.4 |

The entries under the STB and MTB headings are the peak generalization error for nets of the specified size. The italicized STB entries are the STB runs that yielded best performance. The last two columns compare STB and MTB. The first is the percent difference between MTB and the best STB run. Positive percentages indicate MTB performs better. This test is biased towards STB because it compares a single run of MTB on an unoptimized net size with several independent runs of STB that use different random seeds and are able to find near-optimal net size. The last column is the percent difference between MTB and the average STB. Note that on the important steering task, MTB outperforms STB 20–40%.

## 3.2 1D-DOORS

To test multitask backprop on a real problem, we created an object recognition domain similar in some respects to 1D-ALVINN. In 1D-DOORS the main tasks are to locate doorknobs and recognize door types (single or double) in images of doors collected with a robot-mounted camera. Figure 1 shows several door images. As with 1D-ALVINN, the problem was simplified by using horizontal stripes from image, one for the green channel and one for the blue channel. Each stripe is 30 pixels wide (accomplished by applying Gaussian smoothing to the original 150 pixel wide image) and occurs at the vertical location of the doorknob. 10 tasks were used:

- horizontal location of doorknob
- horizontal location of doorway center
- horizontal location of left door jamb
- width of left door jamb
- horizontal location of left edge of door
- single or double door
- width of doorway
- horizontal location of right door jamb
- width of right door jamb
- horizontal location of right edge of door

The difficulty of 1D-DOORS precludes running as exhaustive a set of experiments as with 1D-ALVINN: runs were done only for the two most important and difficult tasks: doorknob location and door type. STB was tested on nets using 6, 24, and 96 hidden units. MTB was tested on a net with 120 hidden units. The results

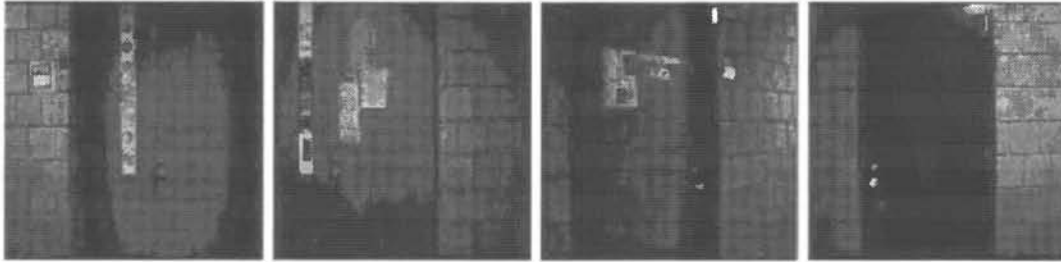

Figure 1: Sample Doors from the 1D-DOORS Domain

are in Table 2. STB generalizes 35–45% worse than MTB on these tasks. Less thorough experiments on the other eight tasks suggest MTB probably always yields performance equal to or better than STB.

Table 2: Performance of STB and MTB on 1D-DOORS.

| TASK | RMS ERROR ON TEST SET | | | | |
|------|------|------|------|------|------|
| | Single Task Backprop | | | MTB | Change MTB |
| | 6HU | 24HU | 96HU | 120HU | to Best STB |
| Doorknob Loc | .085 | .082 | *.081* | .062 | +33.9 % |
| Door Type | .129 | *.086* | .096 | .059 | +45.8 % |

## 4  DISCUSSION

In our experience, multitask backprop usually generalizes better than single task backprop. The few cases where STB has been better is on simpler tasks, and there the difference between MTB and STB was small. Multitask backprop appears to provide the most benefit on hard tasks. MTB also usually learns in fewer epochs than STB. When all tasks must be learned, MTB is computationally more efficient than training single nets. When few tasks are important, however, STB is usually more efficient (but also less accurate).

Tasks do not always learn at the same rate. It is important to watch the training curve of each MTB task individually and stop training each task when its performance peaks. The easiest way to do this to take a snapshot of the net when performance peaks on a task of interest. MTB does not mean one net should be used to predict all tasks, only that all tasks should be trained on one net so they may benefit each other. *Do not treat tasks as one task just because they are being trained on one net!* Balancing tasks (e.g., using different learning rates for different outputs) sometimes helps tasks learn at similar rates, thus maximizing the potential benefits of MTB. Also, because the training curves for MTB are often more complex due to interactions between tasks (MTB curves are frequently multimodal), it is important to train MTB nets until all tasks appear to be overtraining. Restricting the capacity of MTB nets to force sharing or prevent overtraining usually hurts performance instead of helping it. MTB does not depend on restricted net capacity.

We *created* the extra tasks in 1D-ALVINN and 1D-DOORS specifically because we thought they would improve performance on the important tasks. Multitask backprop can be used in other ways. Often the world gives us related tasks to learn. For example, the Calendar Apprentice System (CAP)[4] learns to predict the *Location, Time_Of_Day, Day_Of_Week,* and *Duration* of the meetings it schedules. These tasks are functions of the same data, share many common features,

and would be easy to learn together. Sometimes the world gives us related tasks in mysterious ways. For example, in a medical domain we are examining where the goal is to predict illness severity, half of the lab tests are cheap and routinely measured before admitting a patient (e.g., blood pressure, pulse, age). The rest are expensive tests requiring hospitalization. Users tell us it would be useful to predict if the severity of the illness warrants admission (and further testing) using just the pre-admission tests. Rather than ignore the most diagnostically useful information in the database, we use the expensive tests as additional tasks the net must learn. They are not very predictable from the simple pre-admission tests, but providing them to the net as outputs helps it learn illness severity better. Multitask backprop is one way of providing to a net information that at run time would only be available in the future. The training signals are needed only for the training set because they are outputs—not inputs—to the net.

## 5   RELATED WORK

Training nets with many outputs is not new; NETtalk [9] used one net to learn phonemes and stress. This approach was natural for NETtalk where the goal was to control a synthesizer that needed both phoneme and stress commands at the same time. No analysis, however, was made of the advantages of using one net for all the tasks[3], and the different outputs were not treated as independent tasks. For example, the NETtalk stress task overtrains badly long before the phoneme task is learned well, but NETtalk did not use different snapshots of the net for different tasks. NETtalk also made no attempt to balance tasks so that they would learn at a similar rate, or to add new tasks that might improve learning but which would not be useful for controlling the synthesizer.

Work has been done on serial transfer between nets [8]. Improved learning speed was reported, but not improved generalization. The key difficulties with serial transfer are that it is difficult to scale to many tasks, it is hard to prevent catastrophic interference from erasing what was learned previously, the learning sequence must be defined manually, and serial learning precludes mutual benefit between tasks. This work is most similar to catalytic hints [1][10] where extra tasks correspond to important learnable features of a main task. This work extends hints by showing that tasks can be related in more diverse ways, by expanding the class of mechanisms responsible for multitask backprop, by showing that capacity restriction is not an important mechanism for multitask backprop [2], and by demonstrating that creating many new related tasks may be an efficient way of providing domain-specific inductive bias to backprop nets.

## 6   FUTURE WORK

We used *vanilla* backprop to show the benefit of training many related tasks on one net. Additional techniques may enhance the effects. Regularization and incremental net growing procedures might improve performance by promoting sharing without restricting capacity. New techniques may also be necessary to enhance the benefit of multitask backprop. Automatic balancing of task learning rates would make MTB easier to use. It would also be valuable to know when the different MTB mechanisms are working—they might be useful in different kinds of domains and might benefit from different regularization or balancing techniques. Finally, although MTB usually seems to help and rarely hurts, the only way to know it

helps is to try it. It would be better to have a predictive theory of how tasks should relate to benefit MTB, particularly if new tasks are to be created only to provide a multitask benefit for the other important tasks in the domain.

# 7  SUMMARY

Five mechanisms that improve generalization performance on nets trained on multiple related tasks at the same time have been identified. These mechanisms work without restricting net capacity or otherwise reducing the net's VC-dimension. Instead, they exploit backprop's ability to combine the error terms for related tasks into an aggregate gradient that points towards better underlying representations. Multitask backprop was tested on a simulated domain, 1D-ALVINN, and on a real domain, 1D-DOORS. It improved generalization performance on hard tasks in these domains 20–40% compared with the best performance that could be obtained from multiple trials of single task backprop.

### Acknowledgements

Thanks to Tom Mitchell, Herb Simon, Dean Pomerleau, Tom Dietterich, Andrew Moore, Dave Touretzky, Scott Fahlman, Sebastian Thrun, Ken Lang, and David Zabowski for suggestions that have helped shape this work. This research is sponsored in part by the Advanced Research Projects Agency (ARPA) under grant no. F33615-93-1-1330.

## Footnotes

[1]In these experiments the nets have sufficient capacity to find independent minima for the tasks. They are not forced to share the hidden layer representations. But because the initial weights are random, they do initially share the hidden layer and will separate the tasks (i.e., use independent chunks of the hidden layer for each task) only if learning causes them to.

[2]We have yet to determine how to directly test the hypothesis that representations for $F$ in the intersection of $T$ and $T'$ perform better. Testing this requires interpreting representations learned for *real* tasks; experiments on test problems demonstrate only that search is biased towards the intersection, not that the intersection is the right place to be.

[3]See [5] for evidence that NETtalk is harder to learn using separate nets.

# References

[1] Y.S. Abu-Mostafa, "Learning From Hints in Neural Networks," *Journal of Complexity* **6**:2, pp. 192–198, 1989.

[2] Y.S. Abu-Mostafa, "Hints and the VC Dimension," *Neural Computation,* **5**:2, 1993.

[3] R. Caruana, "Multitask Connectionist Learning," *Proceedings of the 1993 Connectionist Models Summer School,* pp. 372–379, 1993.

[4] L. Dent, J. Boticario, J. McDermott, T. Mitchell, and D. Zabowski, "A Personal Learning Apprentice," *Proceedings of 1992 National Conference on Artificial Intelligence,* 1992.

[5] T.G. Dietterich, H. Hild, and G. Bakiri, "A Comparative Study of ID3 and Backpropagation for English Text-to-speech Mapping," *Proceedings of the Seventh International Conference on Artificial Intelligence,* pp. 24–31, 1990.

[6] G.E. Hinton, "Learning Distributed Representations of Concepts," *Proceedings of the Eight International Conference of The Cognitive Science Society,* pp. 1–12, 1986.

[7] D.A. Pomerleau, "Neural Network Perception for Mobile Robot Guidance," Carnegie Mellon University: *CMU-CS-92-115,* 1992.

[8] L.Y. Pratt, J. Mostow, and C.A. Kamm, "Direct Transfer of Learned Information Among Neural Networks," *Proceedings of AAAI-91,* 1991.

[9] T.J. Sejnowski and C.R. Rosenberg, "NETtalk: A Parallel Network that Learns to Read Aloud," John Hopkins: *JHU/EECS-86/01,* 1986.

[10] S.C. Suddarth and A.D.C. Holden, "Symbolic-neural Systems and the Use of Hints for Developing Complex Systems," *International Journal of Max-Machine Studies* **35**:3, pp. 291–311, 1991.